# Q-Learning with Hidden-Unit Restarting

**Charles W. Anderson**
Department of Computer Science
Colorado State University
Fort Collins, CO 80523

## Abstract

Platt's resource-allocation network (RAN) (Platt, 1991a, 1991b) is modified for a reinforcement-learning paradigm and to "restart" existing hidden units rather than adding new units. After restarting, units continue to learn via back-propagation. The resulting restart algorithm is tested in a Q-learning network that learns to solve an inverted pendulum problem. Solutions are found faster on average with the restart algorithm than without it.

## 1  Introduction

The goal of supervised learning is the discovery of a compact representation that generalizes well. Such representations are typically found by incremental, gradient-based search, such as error back-propagation. However, in the early stages of learning a control task, we are more concerned with fast learning than a compact representation. This implies a local representation with the extreme being the memorization of each experience. An initially local representation is also advantageous when the learning component is operating in parallel with a conventional, fixed controller. A learning experience should not generalize widely; the conventional controller should be preferred for inputs that have not yet been experienced.

Platt's resource-allocation network (RAN) (Platt, 1991a, 1991b) combines gradient search and memorization. RAN uses locally tuned (gaussian) units in the hidden layer. The weight vector of a gaussian unit is equal to the input vector for which the unit produces its maximal response. A new unit is added when the network's error magnitude is large and the new unit's radial domain would not significantly overlap domains of existing units. Platt demonstrated RAN on the supervised learning task

of predicting values in the Mackey-Glass time series.

We have integrated Platt's ideas with the reinforcement-learning algorithm called Q-learning (Watkins, 1989). One major modification is that the network has a fixed number of hidden units, all in a single-layer, all of which are trained on every step. Rather than adding units, the least useful hidden unit is selected and its weights are set to new values, then continue the gradient-based search. Thus, the unit's search is *restarted*. The temporal-difference errors control restart events in a fashion similar to the way supervised errors control RAN's addition of new units.

The motivation for starting with all units present is that in a parallel implementation, the computation time for a layer of one unit is roughly the same as that for a layer with all of the units. All units are trained from the start. Any that fail to learn anything useful are *re*-allocated when needed.

Here the Q-learning algorithm with restarts is applied to the problem of learning to balance a simulated inverted pendulum. In the following sections, the inverted pendulum problem and Watkin's Q-Learning algorithm are described. Then the details of the restart algorithm are given and results of applying the algorithm to the inverted pendulum problem are summarized.

## 2    Inverted Pendulum

The inverted pendulum is a classic example of an inherently unstable system. The problem can be used to study the difficult credit assignment problem that arises when performance feedback is provided only by a failure signal. This problem has often used to test new approaches to learning control (from early work by Widrow and Smith, 1964, to recent studies such as Jordan and Jacobs, 1990, and Whitley, Dominic, Das, and Anderson, 1993). It involves a pendulum hinged to the top of a wheeled cart that travels along a track of limited length. The pendulum is constrained to move within the vertical plane. The state is specified by the position and velocity of the cart and the angle between the pendulum and vertical and the angular velocity of the pendulum.

The only information regarding the goal of the task is provided by the failure signal, or reinforcement, $r_t$, which signals either the pendulum falling past $\pm 12°$ or the cart hitting the bounds of the track at $\pm 1$ m. The state at time $t$ of the pendulum is presented to the network as a vector, $x_t$, of the four state variables scaled to be between 0 and 1.

For further details of this problem and other reinforcement learning approaches to this problem, see Barto, Sutton, and Anderson (1983) and Anderson (1987).

## 3    Q-Learning

The objective of many control problems is to optimize a performance measure over time. For the inverted pendulum problem, we define a *reinforcement signal* to be $-1$ when the pendulum angle or the cart position exceed their bounds, and 0 otherwise. The objective is to maximize the sum of this reinforcement signal over time.

If we had complete knowledge of state transition probabilities we could apply dynamic programming to find the sequence of pushes that maximize the sum of reinforcements. Reinforcement learning algorithms have been devised to learn control strategies when such knowledge is not available. In fact, Watkins has shown that one form of his Q-learning algorithm converges to the dynamic programming solution (Watkins, 1989; Watkins and Dayan, 1992).

The essence of Q-learning is the learning and use of a Q function, $Q(x, a)$, that is a prediction of a weighted sum of future reinforcement given that action $a$ is taken when the controlled system is in a state represented by $x$. This is analogous to the value function in dynamic programming. Specifically, the objective of Q-learning is to form the following approximation:

$$Q(x_t, a_t) \approx \sum_{k=0}^{\infty} \gamma^k r_{t+k+1}$$

where $0 \leq \gamma < 1$ is a discount rate and $r_t$ is the reinforcement received at time $t$.

Watkins (1989) presents a number of algorithms for adjusting the parameters of $Q$. Here we focus on using error back-propagation to train a neural network to learn the $Q$ function. For Q-learning, the following *temporal-difference* error (Sutton, 1988)

$$e_t = r_{t+1} + \gamma \max_{a_{t+1}} [Q(x_{t+1}, a_{t+1})] - Q(x_t, a_t).$$

is derived by using $\max_{a_{t+1}} [Q(x_{t+1}, a_{t+1})]$ as an approximation to $\sum_{k=0}^{\infty} \gamma^k r_{t+k+2}$. See (Barto, Bradtke, and Singh, 1991) for further discussion of the relationships between reinforcement learning and dynamic programming.

## 4   Q-Learning Network

For the inverted pendulum experiments reported here, a neural network with a single hidden layer was used to learn the $Q(x, a)$ function. As shown in Figure 1, the network has four inputs for the four state variables of the inverted pendulum, and two outputs corresponding to the two possible actions for this problem, similar to Lin (1992). In addition to the weights shown, $w$ and $v$, the two units in the output layer each have a single weight with a constant input of 0.5.

The activation function of the hidden units is the approximate gaussian function used by Platt. Let $d_j$ be the squared distance between the current input vector, $x$, and the weights in hidden unit $j$.

$$d_j = \sum_{i=1}^{4} (x_i - w_{j,i})^2$$

Here $x_i$ is the $i^{th}$ component of $x$ at the current time. The output, $y_j$, of hidden unit $j$ is

$$y_j = \begin{cases} \left(1 - \frac{d_j}{\rho}\right)^2, & \text{if } d_j < \rho; \\ 0, & \text{otherwise,} \end{cases}$$

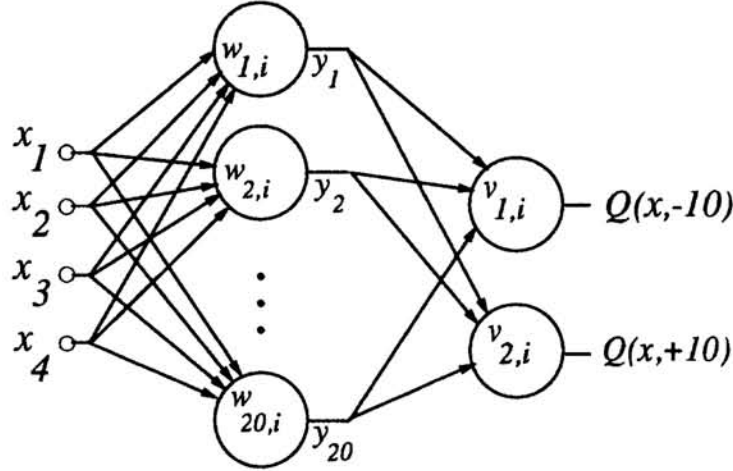

Figure 1: Q-Learning Network

where $\rho$ controls the radius of the region in which the unit's output is nonzero. Unlike Platt, $\rho$ is constant and equal for all units.

The output units calculate weighted sums of the hidden unit outputs and the constant input. The output values are the current estimates of $Q(x_t, -10)$ and $Q(x_t, 10)$, which are predictions of future reinforcement given the current observed state of the inverted pendulum and assuming a particular action will be applied in that state.

The action applied at each step is selected as the one corresponding to the larger of $Q(x_t, -10)$ and $Q(x_t, 10)$. To explore the effects of each action, the action with the lower $Q$ value is applied with a probability that decreases with time:

$$p = \begin{cases} 1 - 0.5\lambda^t, & \text{if } Q(x_t, 10) > Q(x_t, -10); \\ 0.5\lambda^t, & \text{otherwise,} \end{cases}$$

$$a_t = \begin{cases} 10, & \text{with probability } p; \\ -10, & \text{with probability } 1-p. \end{cases}$$

To update all weights, error back-propagation is applied at each step using the following temporal-difference error

$$e_t = \begin{cases} \gamma \max_{a_{t+1}} [Q(x_{t+1}, a_{t+1})] - Q(x_t, a_t), & \text{if failure does not occur on step } t+1, \\ r_{t+1} - Q(x_t, a_t), & \text{if failure occurs on step } t+1. \end{cases}$$

Note that $r_t = 0$ for all non-failure steps and drops out of the first expression.

Weights are updated by the following equations, assuming Unit $j$ is the output unit corresponding to the action taken, and all variables are for the current time $t$.

$$\Delta w_{k,i} = \frac{\beta_h}{\rho} e \, y_k \, v_{j,k} \, (x_i - w_{j,i})$$

$$\Delta v_{j,i} = \beta \, e \, y_i$$

In all experiments, $\rho = 2$, $\lambda = 0.99999$, and $\gamma = 0.9$. Values of $\beta$ and $\beta_h$ are discussed in Section 6.

# 5   Restart Algorithm

After weights are modified by back-propagation, conditions for a restart are checked. If conditions are met, a unit is restarted, and processing continues with the next time step. Conditions and primary steps of the restart algorithm appear below as the numbered equations.

## 5.1   When to Restart

Several conditions must be met before a restart is performed. First, the magnitude of the error, $e_t$, must be larger than usual. To detect this, exponentially-weighted averages of the mean, $\mu$, and variance, $\sigma^2$, of $e_t$ are maintained and used to calculate a normalized error, $e_t'$

$$
\begin{aligned}
e_t' &= e_t - \frac{\mu_t}{(1 - \kappa^t)}, \\
\mu_{t+1} &= \kappa\mu_t + (1 - \kappa)e_t, \\
\sigma_{t+1}^2 &= \kappa\sigma_t^2 + (1 - \kappa)e_t'^2,
\end{aligned}
$$

For our experiments, $\kappa = 0.99$.

Now we can state the first restart condition. A restart is considered on steps for which the magnitude of the error is greater than 0.01 and greater than a constant factor of the error's standard deviation, i.e., whenever

$$
|e_t| > 0.01 \quad \text{and} \quad |e_t| > \alpha\sqrt{\frac{\sigma_t^2}{(1 - \kappa^n)}}. \tag{1}
$$

Of a small number of tested values, $\alpha = 0.2$ resulted in the best performance.

Before choosing a unit to restart for this step, we determine whether or not the current input vector is already "covered" by a unit. Assuming $y_j$ is the output of Unit $j$ for the current input vector, the restart procedure is continued only if

$$
y_j < 0.5, \text{ for } j = 1, \ldots, 20 \tag{2}
$$

## 5.2   Which Unit to Restart

As stated by Mozer and Smolensky (1989), ideally we would choose the least useful unit as the one that results in the largest error when removed from the network. For the Q-network, this requires the removal of one unit at a time, making multiple attempts to balance the pendulum, and determining which unit when removed results in the shortest balancing times. Rather than following this computationally expensive procedure, we simply took the sum of the magnitudes of a hidden unit's output weights as a measure of it's utility. This is one of several utility measures suggested by Mozer and Smolensky and others (e.g., Kloph and Gose, 1969).

After a unit is restarted, it may require further learning experience to acquire a useful function in the network. The amount of learning experience is defined as a sum of magnitudes of the error $e_t$. The sum of error magnitudes since Unit $j$ was

restarted is given by $c_j$. Once this sum surpasses a maximum, $c_{max}$, the unit is again eligible for restarting. Thus, Unit $j$ is restarted when

$$u_j = \min_{j \in \{1,...,20\}} (|v_{1,j}| + |v_{2,j}|) \qquad (3)$$

and

$$c_j > c_{\max}. \qquad (4)$$

Without a detailed search, a value of $c_{\max} = 10$ was found to result in good performance.

### 5.3    New Weights for Restarted Unit

Say Unit $j$ is restarted. It's input weights are set equal to the current input vector, $x$, the one for which the output of the network was in error. One of the two output weights of Unit $j$ is also modified. The output weight through which Unit $j$ modifies the output of the unit corresponding to the action actually taken is set equal to the error, $e_t$. The other output weight is not modified.

$$w_{j,i} = x_i, \text{ for } i = 1,...,4, \qquad (5)$$
$$v_{k,j} = e_t, \qquad (6)$$
$$\text{where } k = \begin{cases} 1, & \text{if } a_t = -10; \\ 2, & \text{if } a_t = 10. \end{cases}$$

## 6    Results

The pendulum is said to be balanced when 90,000 steps (1/2 hour of simulated time) have elapsed without failure. After every failure, the pendulum is reset to the center of the track with a zero angle (straight up) and zero velocities. Performance is judged by the average number of failures before the pendulum is balanced. Averages were taken over 30 runs. Each run consists of choosing initial values for the hidden units' weights from a uniform distribution from 0 to 1, then training the net until the pendulum is balanced for 90,000 steps or a maximum number of 50,000 failures is reached.

To determine the effect of restarting, we ccmpare the performance of the Q-learning algorithm with and without restarts. Back-propagation learning rates are given by $\beta$ for the output units and $\beta_h$ for the hidden units. $\beta$ and $\beta_h$ were optimized for the algorithm without restarts by testing a large number of values. The best values of those tried are $\beta = 0.05$ and $\beta_h = 1.0$. These values were used for both algorithms. A small number of values for the additional restart parameters were tested, so the restart algorithm is not optimized for this problem.

Figure 2 is a graph of the number of steps between failures versus the number of failures. Each algorithm was initialized with the same hidden unit weights. Without restarts the pendulum is balanced for this run after 6,879 failures. With restarts it is balanced after 3,415 failures.

The performances of the algorithms were averaged over 30 runs giving the following results. The restart algorithm balanced the pendulum in all 30 runs, within an

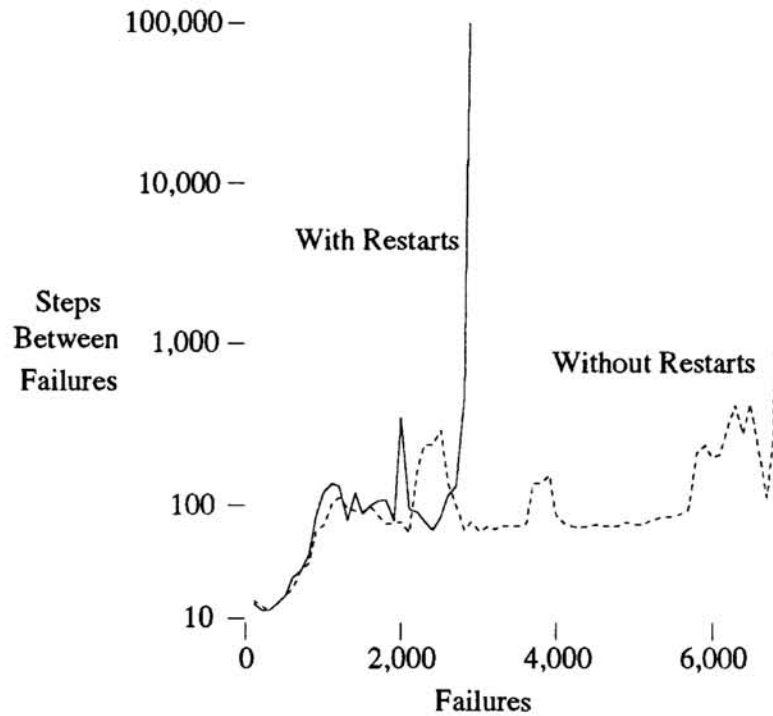

Figure 2: Learning Curves of Balancing Time Versus Failures (averaged over bins of 100 failures)

average of 3,303 failures. The algorithm without restarts was unsuccessful within 50,000 failures for two of the 30 runs. Not counting the unsuccessful runs, this algorithm balanced the pendulum within an average of 4,923 failures. Considering the unsuccessful runs, this average is 7,928 failures.

In studying the timing of restarts, we observe that initially the number of restarts is small, due to the high variance of $e_t$ in the early stages of learning. During later stages, we see that a single unit might be restarted many times (15 to 20) before it becomes more useful (at least according to our measure) than some other unit.

## 7   Conclusion

This first test of an algorithm for restarting hidden units in a reinforcement-learning paradigm led to a decrease in learning time for this task. However, much work remains in studying the effects of each step of the restart procedure. Many alternatives exist, most significantly in the method for determining the utility of hidden units. A significant extension of this algorithm would be to consider units with variable-width domains, as in Platt's RAN algorithm.

### Acknowledgements

The work was supported in part by the National Science Foundation through Grant IRI-9212191 and by Colorado State University through Faculty Research Grant 1-38592.

## References

C. W. Anderson. (1987). Strategy learning with multilayer connectionist representations. Technical Report TR87-509.3, GTE Laboratories, Waltham, MA, 1987. Corrected version of article that was published in Proceedings of the Fourth International Workshop on Machine Learning, pp. 103–114, June, 1987.

A. G. Barto, S. J. Bradtke, and S. P. Singh. (1991). Real-time learning and control using asynchronous dynamic programming. Technical Report 91-57, Department of Computer Science, University of Massachusetts, Amherst, MA, Aug.

A. G. Barto, R. S. Sutton, and C. W. Anderson. (1983). Neuronlike elements that can solve difficult learning control problems. *IEEE Transactions on Systems, Man, and Cybernetics*, 13:835–846. Reprinted in J. A. Anderson and E. Rosenfeld, *Neurocomputing: Foundations of Research*, MIT Press, Cambridge, MA, 1988.

M. I. Jordan and R. A. Jacobs. (1990). Learning to control an unstable system with forward modeling. In D. S. Touretzky, editor, *Advances in Neural Information Processing Systems*, volume 2, pages 324–331. Morgan Kaufmann, San Mateo, CA.

A. H. Klopf and E. Gose. (1969). An evolutionary pattern recognition network. *IEEE Transactions on Systems, Science, and Cybernetics*, 15:247–250.

L.-J. Lin. (1992). Self-improving reactive agents based on reinforcement learning, planning, and teaching. *Machine Learning*, 8(3/4):293–321.

M. C. Mozer and P. Smolensky. (1989). Skeltonization: A technique for trimming the fat from a network via relevance assessment. In D. S. Touretzky, editor, *Advances in Neural Information Systems*, volume 1, pages 107–115. Morgan Kaufmann, San Mateo, CA, 1989.

J. C. Platt. (1991a). Learning by combining memorization and gradient descent. In R. P. Lippmann, J. E. Moody, and D. S. Touretzky, editors, *Advances in Neural Information Processing Systems 3*, pages 714–720. Morgan Kaufmann Publishers, San Mateo, CA.

J. C. Platt. (1991b) A resource-allocating network for function interpolation. *Neural Computation*, 3:213–225.

R. S. Sutton. (1988). Learning to predict by the method of temporal differences. *Machine Learning*, 3:9–44.

C. J. C. H. Watkins. (1989). *Learning with Delayed Rewards*. PhD thesis, Cambridge University Psychology Department.

C. J. C. H. Watkins and P. Dayan. (1992). Q-learning. *Machine Learning*, 8(3/4):279–292.

D. Whitley, S. Dominic, R. Das, and C. Anderson. (1993). Genetic reinforcement learning for neurocontrol problems. *Machine Learning*, to appear.

B. Widrow and F. W. Smith. (1964). Pattern-recognizing control systems. In *Proceedings of the 1963 Computer and Information Sciences (COINS) Symposium*, pages 288–317, Washington, DC. Spartan.
